# Transform-invariant image decomposition with similarity templates

**Chris Stauffer, Erik Miller, and Kinh Tieu**
MIT Artificial Intelligence Lab
Massachusetts Institute of Technology
Cambridge, MA 02139
{*stauffer,emiller,tieu*}*@ai.mit.edu*

## Abstract

Recent work has shown impressive transform-invariant modeling and clustering for sets of images of objects with similar appearance. We seek to expand these capabilities to sets of images of an object class that show considerable variation across individual instances (e.g. pedestrian images) using a representation based on pixel-wise similarities, *similarity templates*. Because of its invariance to the colors of particular components of an object, this representation enables detection of instances of an object class and enables alignment of those instances. Further, this model implicitly represents the regions of color regularity in the class-specific image set enabling a decomposition of that object class into component regions.

## 1  Introduction

Images of a class of objects are often not effectively characterized by a Gaussian distribution or even a mixture of Gaussians. In particular, we are interested in modeling classes of objects that are characterized by similarities and differences between image pixels rather than by the values of those pixels. For instance, images of pedestrians (at a certain scale and pose) can be characterized by a few *regions of regularity* (RORs) such as shirt, pants, background, and head, that have fixed properties such as constant color or constant texture within the region, but tend to be different from each other. The particular color (or texture) of those regions is largely irrelevant. We shall refer to sets of images that fit this general description as *images characterized by regions of regularity*, or ICRORs.

Jojic and Frey [1] and others [2] have investigated transform-invariant modeling and clustering for images of a particular object (e.g., an individual's face). Their method can simultaneously converge on a model and align the data to that model. This method has shown positive results for many types of objects that are effectively modeled by a Gaussian or a mixture of Gaussians. Their work with transformed component analysis (TCA) shows promise for handling considerable variation within the images resulting from lighting or slight misalignments. However, because these models rely on an image set with a fixed mean or mixture of means, they are not directly applicable to ICRORs.

We would also like to address transform-invariant modeling, but use a model which is invariant to the particular color of component regions. One simple way to achieve this is to use edge templates to model *local differences* in image color. In contrast, we have chosen to model *global similarities* in color using a similarity template (ST).

While representations of pixel similarity have previously been exploited for segmentation of single images [3, 4], we have chosen to use them for aggregate modeling of image sets. Similarity templates enable alignment of image sets and decomposition of images into class-specific pixel regions. We note also that registration of two ICRORs can be accomplished by minimizing the mutual information between corresponding pixels [5]. But, there is no obvious way of extending this method to large sets of images without a combinatorial explosion.

Section 2 briefly introduces similarity templates. We investigate their uses for modeling and detection. Section 3 discusses dataset alignment. Section 4 covers their application to decomposing a class-specific set of images into component regions. Future avenues of research and conclusions are discussed Section 5.

## 2   Similarity templates

This section begins with a brief explanation of the similarity template followed by the mechanics of computing and comparing similarity templates. A similarity template $S$ for an $N$-pixel image is an $N$x$N$ matrix. The element $S_{i,j}$ represents the probability that pixel locations $p_i$ and $p_j$ would result from choosing a region and drawing (iid) two samples (pixel locations) from it. More formally,

$$S_{i,j} = \sum_r p(r)p(p_i|r)p(p_j|r), \tag{1}$$

where $p(r)$ is the probability of choosing region $r$ and $p(p_i|r)$ is the probability of choosing pixel location $p_i$ from region $r$.

### 2.1   The "ideal" similarity template

Consider sampling pixel pairs as described above from an $N$-pixel image of a particular object (e.g., a pedestrian) segmented by an oracle into disjoint regions (e.g., shirt, pants, head, feet, background). Assuming each region is equally likely to be sampled and that the pixels in the region are selected with uniform probability, then

$$S_{i,j} = \begin{cases} (\frac{1}{R})(\frac{1}{S_r})^2 & \text{if } r_i = r_j \\ 0 & \text{otherwise,} \end{cases} \tag{2}$$

where $R$ is the number of regions, $S_r$ is the number of pixels in region $r$, and $r_i$ is the region label of $p_i$. If two pixels are from the same region, the corresponding value is the product of the probability $\frac{1}{R}$ of choosing a particular region and the probability $(\frac{1}{S_r})^2$ of drawing that pixel pair. This can be interpreted as a block diagonal co-occurrence matrix of sampled pixel pairs.

In this ideal case, two images of different pedestrians with the same body size and shape would result in the same similarity template regardless of the colors of their clothes, since the ST is a function only of the segmentation. An ST of an image without a pedestrian would exhibit different statistics. Note that even the ST of an image of a blank wall (segmented as a single region) would be different because pixels that are in different regions under the ideal pedestrian ST would be in the same region.

Unfortunately, images do not typically come with labeled regions, and so computation of a similarity template is impossible. However, in this paper, we take advantage of the observation that properties within a region, such as color, are often approximately constant. Using this observation, we can approximate true similarity templates from unsegmented images.

## 2.2 Computing similarity templates

For the purposes of this paper, our model for similarity is based solely on color. Since there is a correlation between color similarity and two pixels being in the same region, we approximate the corresponding value $\tilde{S}_{i,j}$ with a measure of color similarity:

$$\tilde{S}_{i,j} \quad = \quad \frac{1}{NZ_i} \exp\left( \frac{-||I_i - I_j||^2}{\sigma_i^2} \right), \tag{3}$$

where $I_i$ and $I_j$ are pixel color values, $\sigma_i^2$ is a parameter that adjusts the color similarity measure as a function of the pixel color distribution in the image, and $Z_i$ is the sum of the $i^{th}$ row. This normalization is required because large regions have a disproportionate effect on the ST estimate. The choice of $\sigma_i^2$ had little effect on the resulting ST.

If each latent region had a constant but unique color and the regions were of equal size, then as $\sigma_i^2$ approaches zero this process reconstructs the "ideal" similarity template defined in Equation 1. Although region colors are neither constant nor unique, this approximation has proven to work well in practice.

It is possible to add a spatial prior based on the relative pixel location to model the fact that similarities tend to local, but we will rely on the statistics of the images in our data set to determine whether (and to what extent) this is the case. Also, it may be possible to achieve better results using a more complex color model (e.g., hsv with full covariance) or broadening the measure of similarity to include other modalities (e.g., texture, motion, depth, etc.).

Figure 1 shows two views of the same similarity template. The first view represents each pixel's similarity to every other pixel. The second view contains a sub-image for each pixel which highlights the pixels that are most likely produced by the same region. Pixels in the shirt tend to highlight the entire shirt and the pants (to a lesser amount). Pixels in the background tend to be very dissimilar to all pixels in the foreground.

## 2.3 Aggregate similarity templates (AST)

We assume each estimated ST is a noisy measurement of the true underlying joint distribution. Hence we compute an aggregate similarity template (AST) as the mean $\bar{S}$ of the ST estimates over an entire class-specific set of $K$ images:

$$\bar{S}_{i,j} = \frac{1}{K} \sum_{k=1}^{K} \tilde{S}_{i,j}^k. \tag{4}$$

For this quantity to be meaningful, the RORs must be in at least partial correspondence across the training set. Note that this is a less restrictive assumption than assuming edges of regions are in correspondence across an image set, since regions have greater support. Being the mean of a set of probability distributions, the AST is also a valid joint probability distribution.

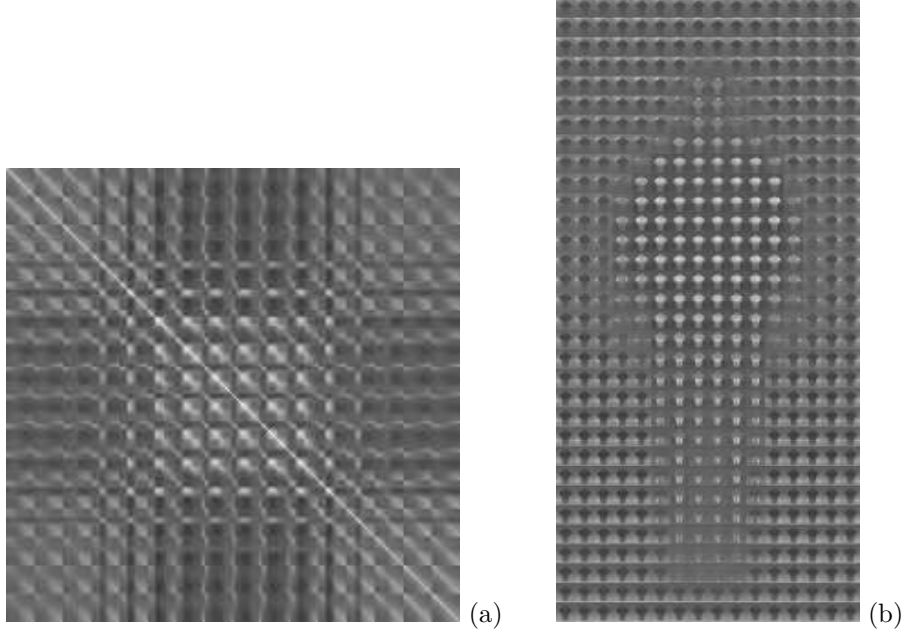

(a)                               (b)

Figure 1: (a) The $N$x$N$ aggregate similarity template for pedestrian data set. (b) An alternate view of (a). This view is a $width^2$x$height^2$ version of (a). Each sub-image represents the row of the original AST that corresponds to that pixel. Each sub-image highlights the pixels that are most similar to the pixel it represents.

## 2.4   Comparing similarity templates

To compare an estimated similarity template $\tilde{S}$ to an aggregate similarity template $\bar{S}$ we evaluate their dot product[1]:

$$s(\bar{S}, \tilde{S}) = \sum_i \sum_j \bar{S}_{i,j} \tilde{S}_{i,j}. \tag{5}$$

We are currently investigating other measures for comparison. By thresholding the ratio of the dot product of a particular image patch under and AST trained on pedestrian image patches versus an AST trained on random image patches, we can determine whether a person is present in the image. In previous work [6], we have illustrated encouraging detection performance.

## 3   Data set alignment

In this paper, we investigate a more difficult problem: alignment of a set of images. To explore this problem, we created a set of 128x64 images of simulated pedestrians. These pedestrians were generated by creating four independently-colored regions corresponding to shirts, pants, head, and background. Each region was given a random color. The RGB components were chosen from a uniform distribution $[0,1]$. Then, independent Gaussian noise was added to each pixel ($\sigma = .1$). Finally the images were translated uniformly up to 25% of the size of the object. Figure 2 shows examples of these images.

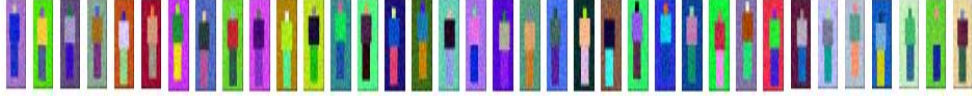

Figure 2: A set of randomly generated "pedestrian" images used in alignment experimetns.

Using the *congealing* procedure of Miller et al. [2], we iteratively estimated the latent variables (translations) that maximized the probability of the image STs to the AST and re-estimated the AST. We were able to align the images to within .5 pixels on average.

## 4   Decomposing the similarity template

This section explains how to derive a factorized representation from the AST that will be useful for recognition of particular instances of a class and for further refinement of detection. This representation is also useful in approximating the template to avoid the $O(N^2)$ storage requirements.

An AST represents the similarity of pixels within an image across an entire class-specific data set. Pairwise statistics have been used for segmentation previously [3]. Recently, work centered on factoring joint distributions has gained increasing attention [7, 8, 9, 10]. Rather than estimating two sets of marginals (conditioned on a latent variable) that explain co-occurrence data (e.g. word-document pairs), we seek a single set of marginals conditioned on a latent variable (the ROR) that explain our co-occurrence data (pixel position pairs). Hence, it is a density factorization in which the two conditional factors are identical (Equation 1). We refer to this as *symmetric factorization* of a joint density.

Also, rather than treating pixel brightness (darkness, redness, blueness, or hue) as a value to be reconstructed in the decomposition, we chose to represent pixel similarity. In contrast to simply treating images as additive mixtures of basis functions [9], our decomposition will get the same results on a database of images of digits written in black on white paper or in white on a black board and color images introduce no difficulties for our methods.

We would like to estimate the factors from Equation 1 that best reconstruct our measured AST, $\bar{S}$. Let $\hat{S}$ be the estimate of $\bar{S}$ constructed from these factors. Given the number of regions $R$, it is possible to estimate the priors for each region $p(r)$ and the probability of each region producing each pixel $p(p_i|r)$. The error function we minimize is the KL-divergence between the empirically measured $\bar{S}$ and our parameterized estimate $\hat{S}$,

$$E = \sum_i \sum_j \bar{S}_{i,j} \log\left(\frac{\bar{S}_{i,j}}{\hat{S}_{i,j}}\right) \tag{6}$$

as in [8]. Because our model $\bar{S}$ is symmetric, this case can be updated with only two rules:

$$p^{new}(p_i|r) \propto p(p_i|r) \sum_{p_j} p(r)p(p_j|r)\frac{\hat{S}(p_i,p_j)}{\bar{S}(p_i,p_j)}, \text{ and} \tag{7}$$

$$p^{new}(r) \propto p(r) \sum_{p_i} \sum_{p_j} p(p_j|r)p(p_i|r)\frac{\hat{S}(p_i,p_j)}{\bar{S}(p_i,p_j)}. \tag{8}$$

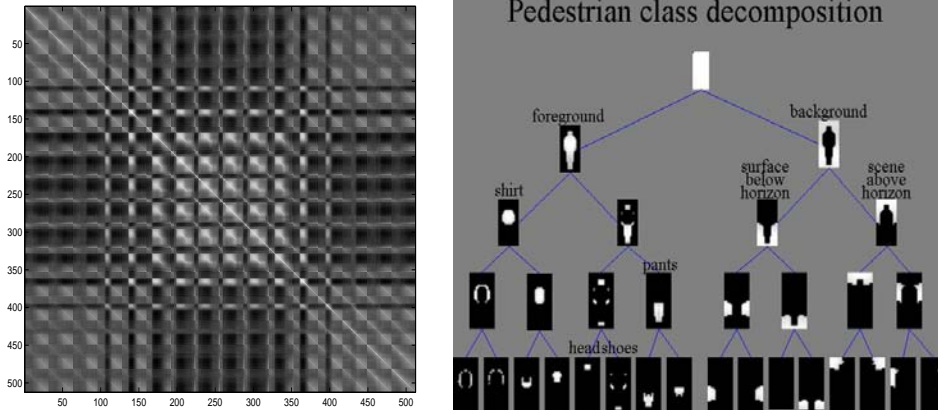

Figure 3: The similarity template and the corresponding automatically generated binary decomposition of the images in the pedestrian data set. The root node represents every pixel in the image. The first branch splits foreground vs. background pixels. Other nodes correspond to shirt, legs, head, and background regions.

The more underlying regions we allow our model, the closer our estimate will approximate the true joint distribution. These region models tend to represent parts of the object class. $p(p_i|r)$ will tend to have high probabilities for a set of pixels belonging to the same region. We take advantage of the fact that aligned pedestrian images are symmetric about the vertical axis by adding a "reflected" aggregate similarity template to the aggregate similarity template. The resulting representation provides a compact approximation of the AST ($O(RN)$ rather than $O(N^2)$).

Rather than performing a straight $R$-way decomposition of the AST to obtain $R$ pixel region models, we extracted a hierarchical segmentation in the form of a binary tree. Given the initial region-conditioned marginals $p(p_i|r_0)$ and $p(p_i|r_1)$, each pixel was assigned to the region with higher likelihood. This was iteratively applied to the ASTs defined for each sub-region. Region priors were set to 0.5 and not adapted in order to encourage a balanced cut.

The probabilistic segmentation can be employed to accumulate robust estimates of statistics of the region. For instance, the mean pixel value can be calculated as a weighted mean where the pixels are weighted by $p(p_i|r)$.

## 4.1 Decomposing pedestrians

Because the data collected at our lab showed limited variability in lighting, background composition, and clothing, we used the MIT CBCL pedestrian data set which contains images of 924 unique, roughly aligned pedestrians in a wide variety of environments to estimate the AST. Figure 3 shows the resulting hierarchical segmentation for the pedestrian AST. Since this intuitive representation was derived automatically with absolutely no knowledge about pedestrians, we hope other classes of objects can be similarly decomposed into RORs.

In our experience, a color histogram of all the pixels within a pedestrian is not useful for recognition and was almost useless for data mining applications. Here we propose a class-conditional color model. It determines a color model over each region that our algorithm has determined contain similar color information within this class of objects. This allows us to obtain robust estimates of color in the regions

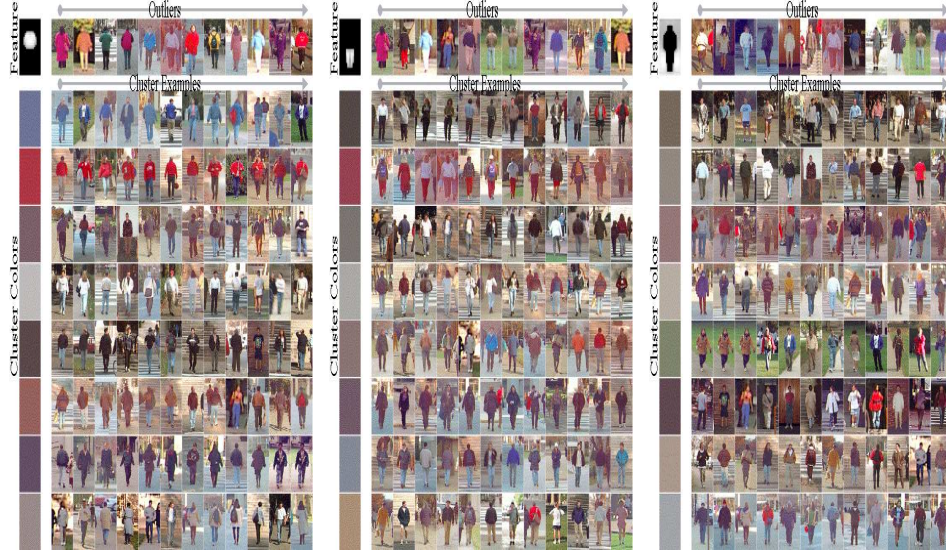

Figure 4: Results of automatic clustering on three components: shirt, pants, and the background. Each shows the feature, the most unusual examples of that region, followed by the 12 most likely examples for the eight prototypical colors of that region.

of regularity. Further, as a result of our probabilistic segmentation, the values of $p(p_i|r)$ indicate which pixels are *most* regular in a region which enables us to weight the contribution of each pixel to the color model.

For the case of pedestrian-conditional color models, the regions roughly correspond to shirt color, pant color, feet color, head color, and some background color regions. The colors in a region *of a single image* can be modeled by color histograms, Gaussians, or mixtures of Gaussians. These region models can be clustered *across images* to determine a density of shirt colors, pant colors, and other region colors within a particular environment. This enables not only an efficient factored color component codebook, but anomaly detection based on particular regions and higher order models of co-occurrences between particular types of regions. To illustrate the effectiveness of our representation we chose the simplest model for the colors in each region–a single Gaussian in RGB space. The mean and variance of each Gaussian was computed by weighting the pixels represented by the corresponding node by $p(p_i|r)$. This biases the estimate towards the "most similar" pixels in the region (e.g., the center of the shirt or the center of the legs). This allows us to represent the colors of each pedestrian image with 31 means and variances corresponding to the $(2^{treeheight} - 1)$ nodes.

We investigated unsupervised clustering on components of the conditional color model. We fit a mixture of eight Gaussians to the 924 color means for each region. Figure 4 shows the 12 pedestrians with the highest probability under each of the eight models and the 12 most unusual pedestrians with respect to that region for three of the nodes of the tree: shirt color, pant color, and color of the background. Red, white, blue, and black shirts represent a significant portion of the database. Blue jeans are also very common in the Boston area (where the CBCL database was collected). Indoor scenes tended to be very dark, and cement is much more common than grass.

# 5 Conclusions

While this representation shows promise, it is not ideal for many problems. First, it is expensive in both memory and computation. Here, we are only using a simple measure of pairwise similarity–color similarity. In the future, similarity templates could be applied to different modalities including texture similarity, depth similarity, or motion similarity.

While computationally intensive, we believe that similarity templates can provide a unified approach to the extraction of possible class-specific targets from an image database, alignment of the candidate images, and precomputation of meaningful features of that class. For the case of pedestrians, it could detect potential pedestrians in a database, align them, derive a model of pedestrians, and extract the parameters for each pedestrian. Once the features are computed, query and retrieval can be done efficiently.

We have introduced a new image representation based on pixel-wise similarity. We have shown its application in both alignment and decomposition of pedestrian images.

## Footnotes

[1]In our experimentation KL-divergence, typically used to compare estimates of distributions, proved less robust.

## References

[1] Jojic, N. and B. J. Frey. "Topographic transformation as a discrete latent variable." In *NIPS 12*, S. A. Solla, T. K. Leen and K.-R. Muller (eds), MIT Press, Cambridge, MA.

[2] Miller, E., N. Matsakis, and P. Viola, (2000) "Learning from One Example Through Shared Densities on Transforms." *CVPR2000*, Vol. 1, pp. 464-471.

[3] Shi, J. and J. Malik. "Normalized Cuts and Image Segmentation," In *CVPR* San juan, Puerto Rico, June 1997.

[4] Boykov, Y., O. Veksler and R. Zabih. Fast Approximate Energy Minimization via Graph Cuts, In *ICCV (99)*, September 1999.

[5] Viola, P. *Alignment by Maximization of Mutual Information.* MIT Artificial Intelligence Lab, Ph.D. Thesis AI-TR #1548, June, 1995.

[6] Stauffer, C. and W.E.L. Grimson. "Similarity templates for detection and recognition," *submitted to CVPR (2001).*

[7] Pereira, F.C., N. Tishby, and L. Lee. "Distributional clustering of English words." In *30th Annual Meeting of the Association for Computational Linguistics*, Columbus, Ohio, pages 183–190, 1993.

[8] Thomas Hofmann, "Probabilistic Latent Semantic Analysis," *UAI (99)*, Morgan Kaufmann Publishers, Inc., San Francisco, 1999.

[9] Lee, D. D. and H. S. Seung. "Learning the parts of objects by non-negative matrix factorization." Nature 401, 788-791 (1999).

[10] Stauffer, C.. "Automatic hierarchical classification using time-based co-occurrences." *CVPR (1999)*, Fort Colins, CO, (June 1999).
